# Nonparametric inference of prior probabilities from Bayes-optimal behavior

**Liam Paninski**[*]
Department of Statistics, Columbia University
liam@stat.columbia.edu;   http://www.stat.columbia.edu/~liam

## Abstract

We discuss a method for obtaining a subject's *a priori* beliefs from his/her behavior in a psychophysics context, under the assumption that the behavior is (nearly) optimal from a Bayesian perspective. The method is nonparametric in the sense that we do not assume that the prior belongs to any fixed class of distributions (e.g., Gaussian). Despite this increased generality, the method is relatively simple to implement, being based in the simplest case on a linear programming algorithm, and more generally on a straightforward maximum likelihood or maximum *a posteriori* formulation, which turns out to be a convex optimization problem (with no non-global local maxima) in many important cases. In addition, we develop methods for analyzing the uncertainty of these estimates. We demonstrate the accuracy of the method in a simple simulated coin-flipping setting; in particular, the method is able to precisely track the evolution of the subject's posterior distribution as more and more data are observed. We close by briefly discussing an interesting connection to recent models of neural population coding.

## Introduction

Bayesian methods have become quite popular in psychophysics and neuroscience (*1–5*); in particular, a recent trend has been to interpret observed biases in perception and/or behavior as optimal, in a Bayesian (average) sense, under ecologically-determined prior distributions on the stimuli or behavioral contexts under study. For example, (*2*) interpret visual motion illusions in terms of a prior weighted towards slow, smooth movements of objects in space.

In an experimental context, it is clearly desirable to empirically obtain estimates of the prior the subject is operating under; the idea would be to then compare these experimental estimates of the subject's prior with the ecological prior he or she "should" have been using. Conversely, such an approach would have the potential to establish that the subject is not behaving Bayes-optimally under any prior, but rather is in fact using a different, non-Bayesian strategy. Such tools would also be quite useful in the context of studies of learning and generalization, in which we would like to track the time course of a subject's adaptation to an experimentally-chosen prior distribution (*5*). Such estimates of the subject's prior have in the past been rather qualitative, and/or limited to simple parametric families (e.g.,

---
[*]We thank N. Daw, P. Hoyer, S. Inati, K. Koerding, I. Nemenman, E. Simoncelli, A. Stocker, and D. Wolpert for helpful suggestions, and in particular P. Dayan for pointing out the connection to neural population coding models. This work was supported by funding from the Howard Hughes Medical Institute, Gatsby Charitable Trust, and by a Royal Society International Fellowship.

the width of a Gaussian may be fit to the experimental data, but the actual Gaussian identity of the prior is not examined systematically).

We present a more quantitative method here. We first discuss the method in the general case of an arbitrarily-chosen loss function (the "cost" which we assume the subject is attempting to minimize, on average), then examine a few special important cases (e.g., mean-square and mean-absolute error) in which the technique may be simplified somewhat. The algorithms for determining the subject's prior distributions turn out to be surprisingly quick and easy to code: the basic idea is that each observed stimulus-response pair provides a set of constraints on what the actual prior could be. In the simplest case, these constraints are linear, and the resulting algorithm is simply a version of linear programming, for which very efficient algorithms exist. More generally, the constraints are probabilistic, and we discuss likelihood-based methods for combining these noisy constraints (and in particular when the resulting maximum likelihood, or maximum *a posteriori*, problem can be solved efficiently via ascent methods, without fear of getting trapped in non-global local maxima). Finally, we discuss Bayesian methods for representing the uncertainty in our estimates.

We should point out that related problems have appeared in the statistics literature, particularly under the subject of elicitation of expert opinion (*6–8*); in the machine learning literature, most recently in the area of "inverse reinforcement learning" (*9*); and in the economics/ game theory literature on utility learning (*10*). The experimental economics literature in particular is quite vast (where the relevance to gambling, price setting, etc. is discussed at length, particularly in settings in which "rational" — expected utility-maximizing — behavior seems to break down); see, e.g. Wakker's recent bibliography (www1.fee.uva.nl/creed/wakker/refs/rfrncs.htm) for further references. Finally, it is worth noting that the question of determining a subject's (or more precisely, an opponent's) priors in a gambling context — in particular, in the binary case of whether or not an opponent will accept a bet, given a fixed table of outcomes vs. payoffs — has received attention going back to the foundations of decision theory, most prominently in the discussions of de Finetti and Savage. Nevertheless, we are unaware of any previous application of similar techniques (both for estimating a subject's true prior and for analyzing the uncertainty associated with these estimates) in the psychophysical or neuroscience literature.

## General case

Our technique for determining the subject's prior is based on several assumptions (some of which will be relaxed below). To begin, we assume that the subject is behaving optimally in a Bayesian sense. To be precise, we have four ingredients: a prior distribution on some hidden parameter $\theta$; observed input (stimulus) data, dependent in some probabilistic way on $\theta$; the subject's corresponding output estimates of the underlying $\theta$, given the input data; and finally a loss function $D(.,.)$ that penalizes bad estimates for $\theta$. The fundamental assumption is that, on each trial $i$, the subject is choosing the estimate $\hat{\theta}_i$ of the underlying parameter, given data $x_i$, to minimize the posterior average error

$$\int p(\theta|x_i)D(\hat{\theta}_i,\theta)d\theta \sim \int p(\theta)p(x_i|\theta)D(\hat{\theta}_i,\theta)d\theta, \tag{1}$$

where $p(\theta)$ is the prior on hidden parameters (the unknown object the experimenter is trying to estimate), and $p(x_i|\theta)$ is the likelihood of data $x_i$ given $\theta$. For example, in the visual motion example, $\theta$ could be the true underlying velocity of an object moving through space, the observed data $x_i$ could be a short, noise-contaminated movie of the object's motion, and the subject would be asked to estimate the true motion $\theta$ given the data $x_i$ and any prior conceptions, $p(\theta)$, of how one expects objects to move. Note that we have also implicitly assumed, in this simplest case, that both the loss $D(.,.)$ and likelihood functions $p(x_i|\theta)$ are known, both to the subject and to the experimenter (perhaps from a preceding set of

"learning" trials).

So how can the experimenter actually estimate $p(\theta)$, given the likelihoods $p(x|\theta)$, the loss function $D(.,.)$, and some set of data $\{x_i\}$ with corresponding estimates $\{\hat{\theta}_i\}$ minimizing the posterior expected loss (1)? This turns out to be a linear programming problem (*11*), for which very efficient algorithms exist (e.g., "linprog.m" in Matlab). To see why, first note that the right hand side of expression (1) is linear in the prior $p(\theta)$. Second, we have a large collection of linear constraints on $p(\theta)$: we know that

$$p(\theta) \geq 0 \quad \forall \theta \tag{2}$$

$$\int p(\theta)d\theta = 1 \tag{3}$$

$$\int p(\theta)p(x_i|\theta)\Big[D(\hat{\theta}_i,\theta) - D(z,\theta)\Big]d\theta \leq 0 \quad \forall z \tag{4}$$

where (2-3) are satisfied by any proper prior distribution and (4) is the maximizer condition (1) expressed in slightly different language. (See also (*10*), who noted the same linear programming structure in an application to cost function estimation, rather than the prior estimation examined here.)

The solution to the linear programming problem defined by (2-4) isn't necessarily unique; it corresponds to an intersection of half-spaces, which is convex in general. To come up with a unique solution, we could maximimize a concave "regularizing" function on this convex set; possible such functions include, e.g., the entropy of $p(\theta)$, or its negative mean-square derivative (this function is strictly concave on the space of all functions whose integral is held fixed, as is the case here given constraint (3)); more generally, if we have some prior information on the form of the priors the subject might be using, and this information can be expressed in the "energy" form

$$P[p(\theta)] \sim e^{q[p(\theta)]},$$

for a concave functional $q[.]$, we could use the log of this "prior on priors" $P$. An alternative solution would be to modify constraint (4) to

$$\int p(\theta)p(x_i|\theta)\Big[D(\hat{\theta}_i,\theta) - D(z,\theta)\Big] \leq -\epsilon \quad \forall z,$$

where we can then adjust the slack variable $\epsilon$ until the contraint set shrinks to a single point. This leads directly to another linear programming problem (where we want to make the linear function $\epsilon$ as large as possible, under the above constraints). Note that for this last approach to work — for the linear programming problem to have a solution — we need to ensure that the set defined by the constraints (2-4) is compact; this basically means that the constraint set (4) needs to be sufficiently rich, which, in turn, means that sufficient data (or sufficiently strong prior constraints) are required. We will return to this point below.

Finally, what if our primary assumption is not met? That is, what if subjects are not quite behaving optimally with respect to $p(\theta)$? It is possible to detect this situation in the above framework, for example if the slack variable $\epsilon$ above is found to be negative. However, a different, more probabilistic viewpoint can be taken. Assume the value of the choice $\hat{\theta}_i$ is optimal under some "comparison" noise, that is,

$$\int p(\theta)p(x_i|\theta)\Big[D(\hat{\theta}_i,\theta) - D(z,\theta)\Big] \leq \sigma\eta_i(z) \quad \forall z,$$

with $\eta_i(z)$ a random variable of scale $\sigma > 0$ (assume $\eta$ to be i.i.d. for now, although this may be generalized). If we assume this decision noise $\eta$ has a log-concave density (i.e., the log of the density is a concave function; e.g., Gaussian, or exponential), then so does

its integral (*12*), and the resulting maximum likelihood problem has no non-global maxima and is therefore solvable by ascent methods. To see this, write the log-likelihood of $(p, \sigma)$ given data $\{x_i, \hat{\theta}_i\}$ as

$$L_{\{x_i, \hat{\theta}_i\}}(p, \sigma) = \sum \log \int_{-\infty}^{u_i(z)} dp(\eta),$$

with the sum over the set of all the constraints in (4) and

$$u_i(z) \equiv \frac{1}{\sigma} \int p(\theta)p(x_i|\theta) \left[ D(\hat{\theta}_i, \theta) - D(z, \theta) \right].$$

$L$ is the sum of concave functions in $u_i$, and hence is concave itself, and has no non-global local maxima in these variables; since $\sigma$ and $p$ are linearly related through $u_i$ (and $(p, \sigma)$ live in a convex set), $L$ has no non-global local maxima in $(p, \sigma)$, either. Once again, this maximum likelihood problem may be regularized by prior information[1], maximizing the *a posteriori* likelihood $L(p) - q[p]$ instead of $L(p)$; this problem is similarly tractable by ascent methods, by the concavity of $-q[.]$ (note that this "soft-constraint" problem reduces exactly to the "hard" constraint problem (4) as the noise $\sigma \to 0$)[2].

Note that the estimated value of the noise scale $\sigma$ plays a similar role to that of the slack variable $\epsilon$, above, with the difference that $\epsilon$ can be much more sensitive to the worst trial (that is, the trial on which the subject behaves most suboptimally); we can use either of these slack variables to go back and ask about how close to optimally the subjects were actually performing — large values of $\sigma$, for example, imply sub-optimal performance. An additional interesting idea is to use the computed value of $\eta$ as a kind of outlier test; $\eta$ large implies the trial was particularly suboptimal.

## Special cases

**Maximum *a posteriori* estimation:** The maximum *a posteriori* (MAP) estimator corresponds to the Hamming distance loss function,

$$D(i, j) = 1(i \neq j);$$

this implies that the constraints (4) have the simple form

$$p(\hat{\theta}_i) - p(z)L(\hat{\theta}_i, z) \geq 0,$$

with $L(\hat{\theta}_i, z)$ defined as the largest observed likelihood ratio for $\hat{\theta}_i$ and $z$, that is,

$$L(\hat{\theta}_i, z) \equiv \max_{x_i} \frac{p(x_i|z)}{p(x_i|\hat{\theta}_i)},$$

with the maximum taken over all $x_i$ which led to the estimate $\hat{\theta}_i$. This setup is perhaps most appropriate for a two-alternative forced choice situation, where the problem is one of classification or discrimination, not estimation.

**Mean-square and absolute-error regression:** Our discussion assumes an even simpler form when the loss function $D(.,.)$ is taken to be squared error, $D(x,y) = (x-y)^2$, or absolute error, $D(x,y) = |x-y|$. In this case it is convenient to work with a slightly different noise model than the classification noise discussed above; instead, we may model the subject's responses as optimal plus estimation noise. For squared-error, the optimal $\hat{\theta}_i$ is known to be uniquely defined as the conditional mean of $\theta$ given $x_i$. Thus we may replace the collection of linear inequality constraints (4) with a much smaller set of linear *equalities* (a single equality per trial, instead of a single inequality per trial per $z$):

$$\int \left( p(x_i|\theta)(\theta - \hat{\theta}_i) \right) p(\theta) d\theta = \sigma \eta_i; \tag{5}$$

the corresponding likelihood, again, has no non-global local maxima if $\eta$ has a log-concave density. In the simplest case of Gaussian $\eta$, the maximum likelihood problem may be solved by standard nonnegative least-squares (e.g., "lsqnonneg" or "quadprog" in Matlab).

In the absolute error case, the optimal $\hat{\theta}_i$ is given by the conditional median of $\theta$ given $x_i$ (although recall that the median is not necessarily unique here); thus, the inequality constraints (4) may again be replaced by equalities which are linear in $p(\theta)$:

$$\int_{-\infty}^{\hat{\theta}_i} p(\theta)p(x_i|\theta) - \int_{\hat{\theta}_i}^{\infty} p(\theta)p(x_i|\theta) = \sigma \eta_i;$$

again, for Gaussian $\eta$ this may be solved via standard nonnegative regression, albeit with a different constraint matrix. In each case, $\eta_i$ retains its utility as an outlier score.

## A worked example: learning the fairness of a coin

In this section we will work through a concrete example, to show how to put the ideas discussed above into practice. We take perhaps the simplest possible example, for clarity: the subject observes some number $N$ of independent, identically distributed coin flips, and on each trial $i$ tells us his/her probability of observing tails on the next trial, given that $t = t(i)$ tails were observed in the first $i$ trials[3]. Here the likelihood functions $p(x_i|\theta)$ take the standard binomial form $p(t(i)|p_{tails}) = \binom{i}{t} p_{tails}^t (1 - p_{tails})^{i-t}$ (note that it is reasonable to assume that these likelihoods are known to the subject, at least approximately, due to the ubiquity of binomial data).

Under our assumptions, the subject's estimates $\hat{p}_{tails,i}$ are given as the posterior mean of $p_{tails}$ given the number of tails observed up to trial $i$. This puts us directly in the mean-square framework discussed in equation (5); we assume Gaussian estimation noise $\eta$, construct a regression matrix $A$ of $N$ rows, with the $i$-th row given by $p(t(i)|p_{tails})(p_{tails} - \hat{p}_{tails,i})$. To regularize our estimates, we add a small square-difference penalty of the form $q[p(\theta)] = \int |dp(\theta)/d\theta|^2 d\theta$. Finally, we estimate

$$\hat{p}(\theta) = \arg \min_{p \geq 0; \int_0^1 p(\theta)d\theta = 1} ||Ap||_2^2 + \epsilon q[p],$$

for $\epsilon \approx 10^{-7}$; this estimate is equivalent to MAP estimation under a (weak) Gaussian prior on the function $p(\theta)$ (truncated so that $p(\theta) \geq 0$), and is computed using quadprog.m.

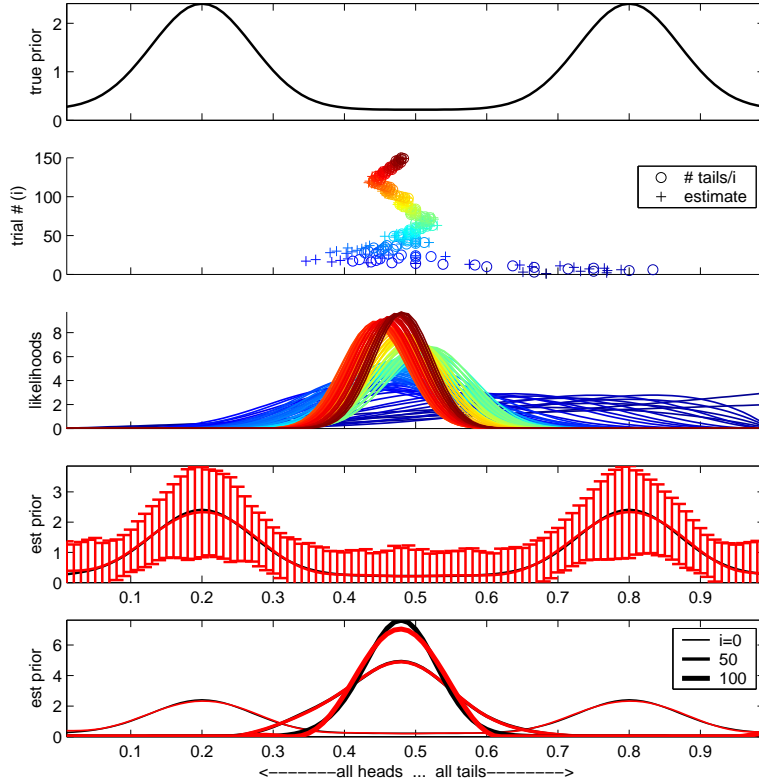

**Figure 1**: Learning the fairness of a coin (numerical simulation). **Top panel**: True prior distribution on coin fairness. The bimodal nature of this prior indicates that the subject expects coins to be unfair (skewed towards heads, $p_{tails} < .5$, or tails, $p_{tails} > .5$) more often than fair ($p_{tails} = .5$). **Second**: Observed data. Open circles indicate the fraction of observed tails $t = t(i)$ as a function of trial number $i$ (the maximum likelihood estimate, MLE, of the fairness and a minimal sufficient statistic for this problem); $+$ symbols indicate the subject's estimate of the coin's fairness, assumed to correspond to the posterior mean of the fairness under the subject's prior. Note the systematic deviations of the subject's estimate from the MLE; these deviations shrink as $i$ increases and the strength of the prior relative to the likelihood term decreases. **Third**: Binomial likelihood terms $\binom{i}{t}p_{tails}^t(1 - p_{tails})^{i-t}$. Color of trace correponds to trial number $i$, as indicated in previous panel (traces are normalized for clarity). **Fourth**: Estimate of prior given 150 trials. Black trace indicates true prior (as in top panel); red indicates estimate $\pm 1$ posterior standard error (computed via importance sampling). **Bottom**: Tracking the evolution of the posterior. Black traces indicate the subject's true posterior after observing 0 (thin trace), 50 (medium trace), and 100 (thick trace) sample coin flips; as more data are observed, the subject becomes more and more confident about the true fairness of the coin ($p = .5$), and the posteriors match the likelihood terms (c.f. third panel) more closely. Red traces indicate the estimated posterior given the full 150 or just the last 100 or 50 trials, respectively (errorbars omitted for visibility). Note that the procedure tracks the evolution of the subject's posterior quite accurately, given relatively few trials.

To place Bayesian confidence intervals around our estimate, we sample from the corresponding (truncated) Gaussian posterior distribution on $p(\theta)$ (via importance sampling with a suitably shifted, rescaled truncated Gaussian proposal density; similar methods are applicable more generally in the non-Gaussian case via the usual posterior approximation techniques, e.g. Laplace approximation). Figs. 1-2 demonstrate the accuracy of the estimated $\hat{p}(\theta)$; in particular, the bottom panels show that the method accurately tracks the evolution of the model subjects' posteriors as an increasing amount of data are observed.

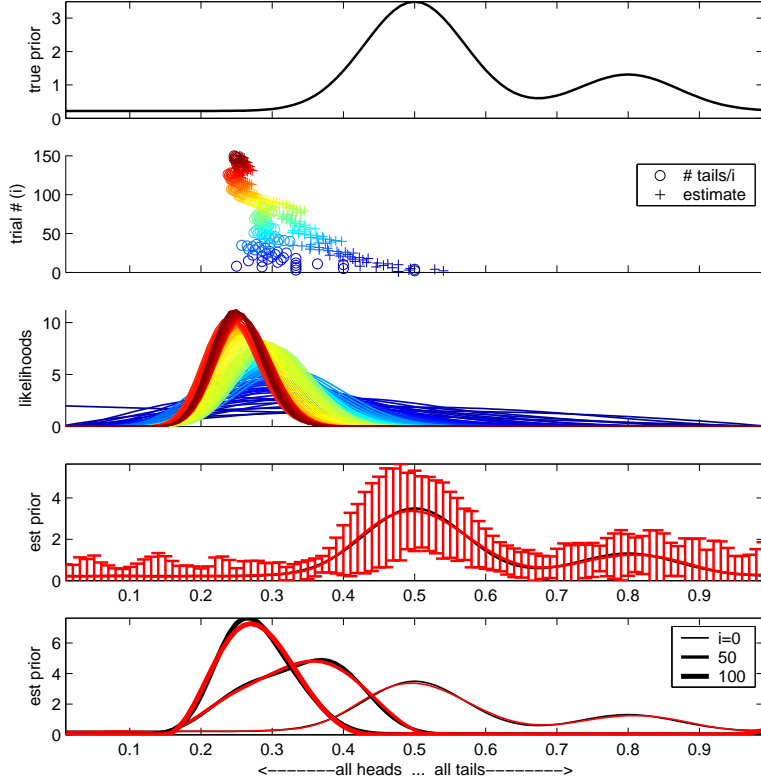

**Figure 2**: Learning an unfair coin ($p_{tails} = .25$). Conventions as in Fig. 1.

## Connection to neural population coding

It is interesting to note a connection to the neural population coding model studied in (*14*) (with more recent work reviewed in (*15*)). The basic idea is that neural populations encode not just stimuli, but probability distributions over stimuli (where the distribution describes the uncertainty in the state of the encoded object). Here the experimentally observed data are neural firing rates, which provide constraints on the underlying encoded "prior" distribution in terms of the individual tuning function of each cell in the observed population.

The simplest model is as follows: the observed spikes $n_i$ from the $i$-th cell are Poisson-distributed, with rate a nonlinear function of a linear functional of some prior distribution,

$$n_i \sim \text{Poiss}\left( g\left( \int p(\theta) f(x_i, \theta) \right) \right),$$

where the kernel $f$ is considered as the cell's "tuning function"; the log-concavity of the likelihood of $p$ is preserved for any nonlinearity $g$ that is convex and log-concave, a class including the linear rectifiers, exponentials, and power-laws (and studied more extensively in (*16*)). Alternately, a simplified model is often used, e.g.:

$$n_i \sim q\left( \frac{n_i - \int p(\theta) f(x_i, \theta)}{\sigma} \right),$$

with $q$ a log-concave density (typically Gaussian) to preserve the concavity of the log-likelihood; in this case, the scale $\sigma$ of the noise does not vary with the mean firing rate,

as it does in the Poisson model. In both cases, the observed firing rates act as constraints oriented linearly with respect to $p$; in the latter case, the noise scale $\sigma$ sets the strength, or confidence, of each such constraint (*2, 3*). Thus, under this framework, given the simultaneously recorded activity of many cells $\{n_i\}$ and some model for the tuning functions $f(x_i, \theta)$, we can infer $p(\theta)$ (and represent the uncertainty in these estimates) using methods quite similar to those developed above.

## Directions

The obvious open avenue for future research (aside from application to experimental data) is to relax the assumptions: that the likelihood and cost function are both known, and that the data are observed directly (without any noise). It seems fair to conjecture that the subject can learn the likelihood and cost functions given enough data, but one would like to test this directly, e.g. by estimating $D(.,.)$ and $p$ together, perhaps under restrictions on the form of $D(.,.)$. As emphasized above, the utility estimation problem has received a great deal of attention, and it is plausible to expect that the methods proposed here for estimation of the prior might be combined with previously-studied methods for utility elicitation and estimation. It is also interesting to consider these elicitation methods in the context of experimental design (*8, 17, 18*), in which we might actively seek stimuli $x_i$ to maximally constrain the possible form of the prior and/or cost function.

## Footnotes

[1]Overfitting here is a symptom of the fact that in some cases — particularly when few data samples have been observed — many priors (even highly implausible priors) can explain the observed data fairly well; in this case, it is often quite useful to penalize these "implausible" priors, thus effectively regularizing our estimates. Similar observations have appeared in the context of medical applications of Markov random field methods (*13*).

[2]Another possible application of this regularization idea is as follows. We may incorporate improper priors — that is, priors which may not integrate to unity (such priors frequently arise in the analysis of reparameterization-invariant decision procedures, for example) — without any major conceptual modification in our analysis, simply by removing the normalization contraint (3). However, a problem arises: the zero measure, $p(\theta) \equiv 0$, will always trivially satisfy the remaining constraints (2) and (4). This problem could potentially be ameliorated by introducing a convex regularizing term (or equivalently, a log-concave prior) on the total mass $\int p(\theta)d\theta$.

[3]We note in passing that this simple binomial paradigm has potential applications to ideal-observer analysis of classical neuroscientific tasks (e.g., synaptic release detection, or photon counting in retina) in addition to potential applications in psychophysics.

## References

1. D. Knill, W. Richards, eds., *Perception as Bayesian Inference* (Cambridge University Press, 1996).

2. Y. Weiss, E. Simoncelli, E. Adelson, *Nature Neuroscience* **5**, 598 (2002).

3. Y. Weiss, D. Fleet, *Statistical Theories of the Cortex* (MIT Press, 2002), chap. Velocity likelihoods in biological and machine vision, pp. 77–96.

4. D. Kersten, P. Mamassian, A. Yuille, *Annual Review of Psychology* **55**, 271 (2004).

5. K. Koerding, D. Wolpert, *Nature* **427**, 244 (2004).

6. R. Hogarth, *Journal of the American Statistical Association* **70**, 271 (1975).

7. J. Oakley, A. O'Hagan, *Biometrika* **under review** (2003).

8. P. Garthwaite, J. Kadane, A. O'Hagan, *Handbook of Statistics* (2004), chap. Elicitation.

9. A. Ng, S. Russell, *ICML-17* (2000).

10. J. Blythe, *AAAI02* (2002).

11. G. Strang, *Linear algebra and its applications* (Harcourt Brace, New York, 1988).

12. Y. Rinott, *Annals of Probability* **4**, 1020 (1976).

13. M. Henrion, *et al.*, Why is diagnosis using belief networks insensitive to imprecision in probabilities?, *Tech. Rep. SMI-96-0637*, Stanford (1996).

14. R. Zemel, P. Dayan, A. Pouget, *Neural Computation* **10**, 403 (1998).

15. A. Pouget, P. Dayan, R. Zemel, *Annual Reviews of Neuroscience* **26**, 381 (2003).

16. L. Paninski, *Network: Computation in Neural Systems* **15**, 243 (2004).

17. K. Chaloner, I. Verdinelli, *Statistical Science* **10**, 273 (1995).

18. L. Paninski, *Advances in Neural Information Processing Systems* **16** (2003).
